# Expected and Unexpected Uncertainty: ACh and NE in the Neocortex

**Angela Yu**     **Peter Dayan**

Gatsby Computational Neuroscience Unit
17 Queen Square, London WC1N 3AR, United Kingdom.
feraina@gatsby.ucl.ac.uk     dayan@gatsby.ucl.ac.uk

## Abstract

Inference and adaptation in noisy and changing, rich sensory environments are rife with a variety of specific sorts of variability. Experimental and theoretical studies suggest that these different forms of variability play different behavioral, neural and computational roles, and may be reported by different (notably neuromodulatory) systems. Here, we refine our previous theory of acetylcholine's role in cortical inference in the (oxymoronic) terms of *expected uncertainty,* and advocate a theory for norepinephrine in terms of *unexpected uncertainty.* We suggest that norepinephrine reports the radical divergence of bottom-up inputs from prevailing top-down interpretations, to influence inference and plasticity. We illustrate this proposal using an adaptive factor analysis model.

## 1   Introduction

Animals negotiating rich environments are faced with a set of hugely complex inference and learning problems, involving many forms of variability. They can be *unsure* which context presently pertains, cues can be systematically more or less *reliable,* and relationships amongst cues can *change* smoothly or abruptly. Computationally, such different forms of variability need to be represented, manipulated, and wielded in different ways. There is ample behavioral evidence that can be interpreted as suggesting that animals do make and respect these distinctions,[5] and there is even some anatomical, physiological and pharmacological evidence as to which neural systems are engaged.[29]

Perhaps best delineated is the involvement of neocortical acetylcholine (ACh) in uncertainty. Following seminal earlier work,[11, 14] we suggested[6, 35] that ACh reports on the uncertainty associated with a top-down model, and thus controls the integration of bottom-up and top-down information during inference. A corollary is that ACh should also control the way that bottom-up information influences the learning of top-down models. Intuitively, this cholinergic signal reports on *expected uncertainty,* such that ACh levels are high when top-down information is not expected to support good predictions about bottom-up data and should be modified according to the incoming data.

We[6, 35] formally demonstrated the inference aspects of this idea using a hidden Markov model (HMM), in which top-down uncertainty derives from slow contextual changes. In extending this quantitative model to learning, we found, surprisingly, that it violated our qualitative theory of ACh. That is, in the HMM model, greater uncertainty in the top-down model (*ie* a lower posterior responsibility for the predominant context), reported by higher ACh levels, leads to comparatively *slower* learning about that context. By contrast, we had expected that higher ACh should lead to faster learning, since it would indicate

that the top-down model is potentially inadequate. In resolving this conflict, we realized that, at least in this particular HMM framework, we had incorrectly fused different sorts of uncertainty. As a further consequence, by thinking more generally about contextual change, we also realized the formal need for a signal reporting on *unexpected uncertainty,* that is, on strong violation of top-down predictions that are expected to be correct. There is suggestive empirical evidence that one of many roles for neocortical norepinephrine (NE) is reporting this;[29] it is also consonant with various existing theories associated with NE.

In sum, we suggest that expected and unexpected uncertainty play complementary but distinct roles in representational inference and learning. Both forms of uncertainties are postulated to decrease the influence of top-down information on representational inference and increase the rate of learning. However, unexpected uncertainty rises whenever there is a global change in the world, such as a context change, while expected uncertainty is a more subtle quantity dependent on internal representations of properties of the world. Here, we start by outlining some of the evidence for the individual and joint roles of ACh and NE in uncertainty. In section 3, we describe a simple, adaptive, factor analysis model that clarifies the uncertainty notions. Differential effects induced by disrupting ACh and NE are discussed in Section 4, accompanied by a comparison to impairments found in animals.

## 2  ACh and NE

ACh and NE are delivered to the cortex from a small number of subcortical nuclei: NE originates solely in the locus coeruleus, while the primary sources of ACh are nuclei in the basal forebrain (nucleus basalis magnocellularis, mainly targeting the neocortex, and medial septum, mainly targeting the hippocampus). Cortical innervations of these modulators are extensive, targeting all cortical regions and layers.[9,30]

As is typical for neuromodulators, physiological studies indicate that the effects of direct application of ACh or NE are confusingly diverse. Within a small cortical area, iontophoresis or perfusion of ACh or NE (or their agonists) may cause synatic facilitation or suppression, depending on the cell and depending on whether the firing is spontaneous or stimulus-evoked; it may also induce direct hyperpolarization or depolarization.[9,10,17] Direct application of either neuromodulator or its agonist, paired with sensory stimulation, results in a general enhancement of stimulus-evoked responses, as well as an increased propensity for experience-dependent reorganization of cortical maps (in contrast, depletion of either substance attenuates cortical plasticity).[9] More interestingly, ACh and NE both seem to selectively suppress intracortical and feedback synaptic transmission while enhancing thalamocortical processing.[8,12,13,15,17,18,20] Based on these roughly similar anatomical and physiological properties, cholinergic and noradrenergic systems have been attributed correspondingly similar general computational roles, such as modulating the signal-to-noise ratio in sensory processing.[9,10]

However, the effects of ACh and NE depletion in animal behavioral studies, as well as microdialysis of the neuromodulators during different conditions, point to more specific and distinct computational roles for ACh and NE. In our previous work on ACh,[6,35] we suggested that it reports on expected uncertainty, *ie* uncertainty associated with estimated parameters in an internal model of the external world. This is consistent with results from animal conditioning experiments, in which animals learn faster about stimuli with variable predictive consequences.[24] A series of lesion studies indicates cortical ACh innervation is essential for this sort of faster learning.[14]

In contrast to ACh, a large body of experimental data associates NE with the specific ability to learn new underlying relationships in the world, especially those contradicting existent knowledge. Locus coeruleus (LC) neurons fire phasically and robustly to novel objects encountered during free exploration,[34] novel sensory stimuli,[25,28] unpredicted changes in stimulus properties such as presentation time,[2] introduction of association of a stimulus

with reinforcement,[19, 28, 32] and extinction or reversal of that association.[19, 28] Moreover, this activation of NE neurons habituates rapidly when there is no predictive value or contingent response associated with the stimuli, and also disappears when conditioning is expressed at a behavioral level.[28]

There are few sophisticated behavioral studies into the interactions between ACh and NE. However, it is known that NE and ACh both rise when contingencies in an operant conditioning task are changed, but while NE level rapidly habituates, ACh level is elevated in a more sustained fashion.[3, 28] In a task designed to tax sustained attention, lesions of the basal forebrain cholinergic neurons induced persistent impairments,[22] while deafferentation of cortical adrenergic inputs did not result in significant impairment compared to controls.[21]

One of the best worked-out computational theories of the drive and function of NE is that of Aston-Jones, Cohen and their colleagues.[1, 33] They studied NE in the context of vigilance and attention in well-learned tasks, showing how NE neurons are driven by selective task-relevant stimuli, and that, influenced by increased electrotonic coupling in the locus coeruleus, a transition from a high tonic, low phasic activity mode to a low tonic, high phasic activity mode is associated with increased behavioral performance through NE's suggested effect of increasing the signal to noise ratio of target cortical cells. This is a very impressive theory, with neural and computational support. However, its focus on well-learned tasks, means that other drives of NE activity (particularly novelty) and effects (particularly plasticity) are downplayed, and a link to ACh is only a secondary concern. We focus on these latter aspects, proposing that NE reports unexpected uncertainty, *ie* uncertainty induced by a mismatch between prediction and observation, such as when there is a dramatic change in the external environment. We do not claim that this is the only role of NE; but do see it as an important complement to other suggestions.

## 3   Inference and Learning in Adaptive Factor Analysis

Our previous model of the role of ACh in cortical inference involved a generative scheme with a discrete contextual variable $z_t$, evolving over time $t$ with slow Markov dynamics $P[z_t = i | z_{t-1} = j] = \mathcal{K}_{ij}$, a discrete representational variable $y_t$ that was stochastically determined by $z_t$, and a noisy observed variable $\mathbf{x}_t \sim \mathcal{N}[\mathbf{y}_{y_t}, \mathcal{I}]$ (normal distribution). The inferential task was to determine $y_t | \mathbf{x}_1, \mathbf{x}_2 \ldots \mathbf{x}_t$; the HMM structure makes this interesting because top-down ($z_t$) and bottom-up ($\mathbf{x}_t$) information have to be integrated. Top down information can be uncertain, in which case mainly bottom-up information $\mathbf{x}_t$ should be used to infer $y_t$. We suggested that ACh reports the uncertainty in the top-down context, namely $1 - \tilde{P}[z_t^* | \mathbf{x}_1 \ldots \mathbf{x}_{t-1}]$, where $z_t^*$ is the most likely value of the context and $\tilde{P}$ indicates the use of an approximation. ACh thereby reports expected uncertainty, as in the qualitative picture above, and appropriately controls cortical inference. However, if one also considers learning, for instance if $P[y_t | z_t]$ is unknown, then the less certain the animal is that $z_t^*$ is the true contextual state, the *less* learning accorded to $P[y_t | z_t^*]$. This is exactly the opposite of what we should expect according to our empirically-supported arguments above.

In fact, this way of viewing ACh is also not consistent with a more systematic reading[5, 16] of Holland & Gallagher's cholinergic results,[14] which imply that ACh is better seen as a report of uncertainty in *parameters* rather than uncertainty in *states*. In order to model this more fitting picture of ACh, we need an explicit model of parameter uncertainty. We constrain the problem to a single, implicit, context $z_t = 1$. It is easiest (and perhaps more realistic) to develop the new picture in a continuous space, in which the parameter governing the relationship between $z_t = 1$ and $y_t$ is $\mu_t$ (scalar for convenience), which is imperfectly known (hence the parameter uncertainty, reported by ACh), and indeed can change. Again, $y_t$ stochastically specifies $\mathbf{x}_t$ through a normal distribution.

Specifying how $\mu_t$ can change over time requires making an assumption about the nature of the context. In particular, novelty plays a critical role in model evolution. In general,

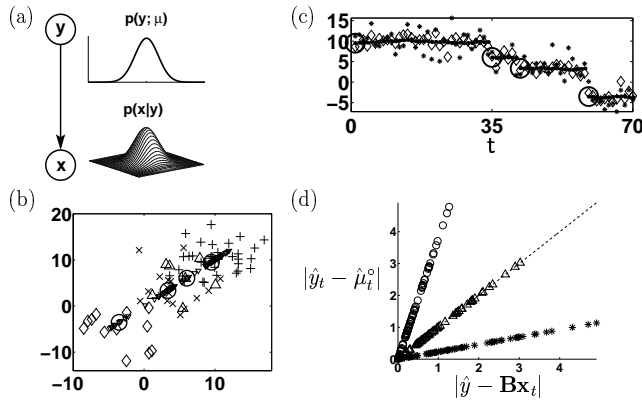

**Figure 1**: Adaptive factor analysis model. (a) 2-layer adaptive factor analysis model, as specified by Eq. 1 & 2. (b) Sample sequence of 70 data points generated with parameters: $q = 0.05$, $\boldsymbol{\xi} = [1; 1]$, $\sigma_\eta = 8$, $\sigma_y = 1$, $\Sigma_x = 9I$, $\sigma_o = 8$. 4 major shifts in $\mu$ occurred (including initial $\mu_1$), whose projections into $\mathbf{x}$ space, $\boldsymbol{\xi}\mu$, are denoted as large circles. +: $\mathbf{x}|\mu \approx 10$, $\triangle$: $\mathbf{x}|\mu \approx 6$, $x$: $\mathbf{x}|\mu \approx 4$, $\lozenge$: $\mathbf{x}|\mu \approx -4$. Small $\nabla$ denotes $y_t$ projected into $\mathbf{x}$ space and fall along the line $\boldsymbol{\xi}y$. (c) Same sequence viewed in $y$ space. $\circ$: major shifts in $\mu$, $\bullet$: $\mu_t$, $\lozenge$: $y_t$, $*$: $\mathbf{x}$ optimally projected into $y$ space, *ie* $\mathbf{Bx} = (\boldsymbol{\xi}^T\Sigma_x^{-1}\boldsymbol{\xi})^{-1}\boldsymbol{\xi}^T\Sigma_x^{-1}\mathbf{x}$, where $\mathbf{Bx}$ is the mean of the posterior distribution of $y$ given only the observation $\mathbf{x}$ and flat priors. (d) Scatter plot of $|\hat{y}_t - \hat{\mu}_t^\circ|$ vs. $|\hat{y}_t - \mathbf{Bx_t}|$. $\circ$:$V_t^\circ = 16$, $*$:$V_t^\circ = 0.04$, $\triangle$:$V_t^\circ = 3.5$, dashed line denotes parity. Larger $V_t^\circ$ corresponds to greater reliance on $\mathbf{x}_t$ rather than $\hat{\mu}_t^\circ$ for inferring $\hat{y}_t$, while the intermediate value of $V_t^\circ = 3.5$ exactly balances top-down uncertainty with bottom-up uncertainty in the inference of $\hat{y}_t$.

we might expect small amounts of novelty, as models continually readjust, and we can allow for this by modeling continual small changes in $\mu_t$. However, in order to allow for the possibility of macroscopic changes implied by substantial novelty (as reported by NE), which are of evident importance in many experiments, we must add a specific component to the model. The interaction between microscopic and macroscopic novelty is essentially the interaction between ACh and NE. In all, assume that

$$\mathbf{x}_t \sim \mathcal{N}[\boldsymbol{\xi}y_t, \Sigma_x] \qquad y_t \sim \mathcal{N}[\mu_t, \sigma_y^2] \qquad \mu_t = \mu_{t-1} + \eta_t + a_t\epsilon_t \qquad (1)$$

$$\eta_t \sim \mathcal{N}[0, \sigma_\eta^2] \qquad \epsilon_t \sim \mathcal{N}[0, \sigma_\epsilon^2] \qquad P[a_t=1] = 1 - P[a_t=0] = q \qquad (2)$$

with the initial value $\mu_0 \sim \mathcal{N}[0, \sigma_o^2]$ (see Figure 1). We will see later that the binary $a_t$ is the key to the model of NE; it comes from an assumption that there can occasionally ($q \ll 1$) be dramatic changes in a model that force its radical revision. $\boldsymbol{\xi}$ is another parameter; we assume it is known and fixed. Figure 1(b) & (c) shows a sample sequence of a particular setting of the model: the output $\mathbf{x}$ can be quite noisy, although there are clear underlying regularities in $y$.

At time $t$, consider the case that we can make the approximation that $\mu_t^\circ \sim \mathcal{N}[\hat{\mu}_t^\circ, V_t^\circ]$, where $\hat{\mu}_t^\circ$ is the estimate of $\mu_t$ and $V_t^\circ$ is its variance (uncertainty), which is reported by ACh. Here, the open circles indicate that this estimate is made *before* $\mathbf{x}_t$ is observed. We first consider how the ACh term influences inference about $y_t$; then go on to study learning. For inference, it can easily be shown that $y_t \sim \mathcal{N}[\hat{y}_t, \hat{\sigma}_t^2]$, where

$$\hat{\sigma}_t^{-2} = (\sigma_y^2 + V_t^\circ)^{-1} + \boldsymbol{\xi}^T\Sigma_x^{-1}\boldsymbol{\xi} \qquad \hat{y}_t = \hat{\sigma}_t^2\left((\sigma_y^2 + V_t^\circ)^{-1}\hat{\mu}_t^\circ + \boldsymbol{\xi}^T\Sigma_x^{-1}\mathbf{x}_t\right) \qquad (3)$$

whence the effect of ACh is exactly as in our qualitative picture. The more uncertainty (*ie* the larger $V_t^\circ$), the smaller the role of the top-down expectation $\hat{\mu}_t^\circ$ in determining $\hat{y}_t$. Examples of just such effects can be found in Figure 1 (d).

For learning, start with the distribution of $\mu_1$ given $\mathbf{x}_1$ and assume $a_1 = 0$. In this case, writing $\Psi = \boldsymbol{\xi}\boldsymbol{\xi}^T\sigma_y^2 + \Sigma_x$, we get

$$p[\mu_1|\mathbf{x}_1, a_1=0] = \mathcal{N}[\boldsymbol{\xi}^T\Psi^{-1}\mathbf{x}_1/\boldsymbol{\xi}^T\Psi^{-1}\boldsymbol{\xi}, 1/\boldsymbol{\xi}^T\Psi^{-1}\boldsymbol{\xi}] \times \mathcal{N}[0, V_0 + \sigma_\eta^2]$$

with the obvious semantics for the product of two Gaussian distributions. This is almost exactly the standard form for a Kalman filter update for $\mu$, and leads to standard results, such as variance of the estimate going initially like $1/\sqrt{t}$, but ultimately reaching an asymptote which balances the rate of change from $\sigma_\eta^2$ and the rate of new information from the $\mathbf{x}_t$. Importantly, in this simple model, the uncertainty in $\mu_t$ does not depend on the prediction errors $\mathbf{x}_t - \boldsymbol{\xi}\hat{\mu}_t$, but rather changes as a function only of time.

However, if one takes into account the possibility that $a_1 = 1$, then the posterior distribution for $\mu_1$ is the two-component mixture

$$p[\mu_1|\mathbf{x}_1] = P[a_1 = 0]p[\mu_1|\mathbf{x}_1, a_1 = 0] + P[a_1 = 1]p[\mu_1|\mathbf{x}_1, a_1 = 1] \qquad (4)$$
$$\propto \mathcal{N}[\boldsymbol{\xi}^T\boldsymbol{\Psi}^{-1}\mathbf{x}_1/\boldsymbol{\xi}^T\boldsymbol{\Psi}^{-1}\boldsymbol{\xi}, 1/\boldsymbol{\xi}^T\boldsymbol{\Psi}^{-1}\boldsymbol{\xi}] \times ((1-q)\mathcal{N}[0, V_0 + \sigma_\eta^2] + q\mathcal{N}[0, V_0 + \sigma_\eta^2 + \sigma_\epsilon^2])$$

As $t$ increases, the number of mixture components in the posterior distribution increases exponentially as $2^t$, since each setting of the $t-$length binary string $a_1a_2 \ldots a_t$ is, barring probability zero accidents, associated with a different component in the mixture. Thus, just as for switching state-space models,[7] exact inference is impractical.

One possibility would be to use a variational approximations.[7,23] From the neural perspective of the involvement of neuromodulators, we propose an approximate learning algorithm in which signals reporting uncertainty, corresponding to our conceptual roles for ACh and NE, control the interactions between the (approximate) distribution at $t-1$, $\tilde{p}[\mu_{t-1}|\mathcal{D}_{t-1}]$, where $\mathcal{D}_{t-1} = \{\mathbf{x}_1, \mathbf{x}_2, \ldots, \mathbf{x}_{t-1}\}$, and bottom-up information relayed by the new observation, $p[y_t|\mathbf{x}_t]$. To control the exponential expansion in the hidden space, we approximate the posterior $\tilde{p}[\mu_{t-1}|\mathcal{D}_{t-1}]$ as a single Gaussian, $\mu_{t-1} \sim \mathcal{N}(\hat{\mu}_{t-1}, V_{t-1})$. $\hat{\mu}_{t-1}$ is our best estimate of $\mu_{t-1}$ after observing $\mathbf{x}_{t-1}t$, and $V_{t-1}$, corresponding to the ACh level, is the uncertainty in our estimate $\hat{\mu}_{t-1}$. In general, we might consider the NE level $\beta_t$ as reporting the posterior responsibility of the $a_t = 1$ component of the equivalent mixture of equation 4. Even more straightforwardly, we can measure a Z-score, namely prediction error scaled by uncertainty in our estimates: $\beta_t = (\hat{\mathbf{x}}_t - \mathbf{x}_t)^T\boldsymbol{\Psi}_t^{-1}(\hat{\mathbf{x}}_t - \mathbf{x}_t)$, where $\hat{\mathbf{x}}_t = \boldsymbol{\xi}\hat{\mu}_{t-1}$ and $\boldsymbol{\Psi}_t = \boldsymbol{\xi}\boldsymbol{\xi}^T(V_{t-1} + \sigma_y^2 + \sigma_\eta^2) + \Sigma_x$, assuming that $a_t = 0$. Whenever $\beta_t$ exceeds a threshold value $\gamma$, ie $\mathbf{x}_t$ is unlikely to have come from an unmodified version of the current component, we assume $\hat{a}_t = 1$. Otherwise, $\hat{a}_t = 0$. Now the learning problem reduces to a modified version of Kalman filter:

$$V_t^\circ = V_{t-1} + \sigma_\eta^2 + Q_t \qquad \text{prediction variance about } \mu_t \qquad (5)$$
$$\mathbf{K}_t = V_t^\circ\boldsymbol{\xi}^T(\boldsymbol{\xi}V_t^\circ\boldsymbol{\xi}^T + \boldsymbol{\xi}\boldsymbol{\xi}^T\sigma_y^2 + \Sigma_x)^{-1} \qquad \text{Kalman gain} \qquad (6)$$
$$V_t = V_t^\circ - \mathbf{K}_{t-1}\boldsymbol{\xi}V_t^\circ \qquad \text{correction variance} \qquad (7)$$
$$\hat{\mu}_t = \hat{\mu}_{t-1} + \mathbf{K}_t(\mathbf{x}_t - \boldsymbol{\xi}\hat{\mu}_{t-1}) \qquad \text{estimated mean} \qquad (8)$$

The difference from the conventional Kalman filter is the additional component of the transition noise variance, $Q_t$, which depends on $\hat{a}_t$: $Q_t = 0$ if $\hat{a}_t = 0$, $Q_t = \sigma_\epsilon^2$ if $\hat{a}_t = 1$. Closer examination indicates that the ACh ($V_t$) and NE ($\beta_t$) signals have the desired semantics. In the learning algorithm, large uncertainty about the mean estimate, $V_t$, results in large Kalman gain, $\mathbf{K}_t$, which causes a large shift in $\mu_{t+1}$. Large $V_t$ also weakens the influence of top-down information in inference as in equation 3. High NE levels also leads to faster learning: large $\beta_t$ means $\hat{a}_t = 1$, which causes $Q_t = \sigma_\mu$ (rather than $Q_t = 0$ had $\hat{a}_t$ been 0), ultimately resulting in a large Kalman gain and thus fast shifting of $\hat{\mu}$. High NE levels also enhances the dominance of bottom-up information in inference via its interactions with ACh: large $\beta_t$ promotes large $V_t$. Note that this system predicts interesting reciprocal relationships between ACh and NE: higher ACh leads to smaller normalized prediction errors and therefore less active NE signalling, whereas greater NE would generally increase estimator uncertainty and thus ACh level.

Figure 2(a) shows an example sequence of $\mu_1, \mu_2, \ldots$ generated from a model (same parameters as in Figure 1), and the estimated means using our approximate learning algorithm. The learning algorithm is clearly able to adjust to major changes in $\mu_t$, although

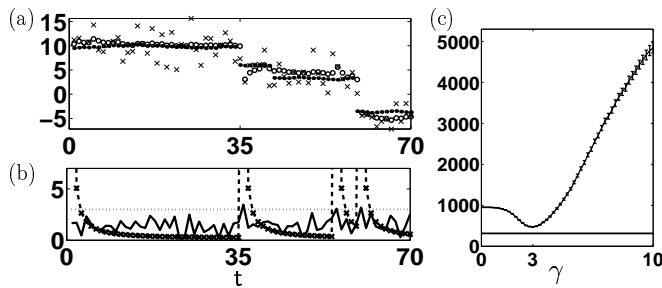

**Figure 2**: Approximate learning algorithm. (a) x: $\mathbf{x}_t$ projected into $y$ space, $\bullet$: actual $\mu_t$, $\circ$: estimated means $\hat{\mu}_t$. General patterns of $\mu_t$ are captured by $\hat{\mu}_t$, though details may differ. $\gamma = 3$. (b) $-*$ $-$: ACh, $-$: NE, $\cdots$: $\gamma$. ACh level rises whenver $\hat{a}_t$ detected to be 1 (NE level exceeds $\gamma$) and then smoothly falls. NE level is constant monitor of prediction error. (c) Mean summed square error over 200-step sequences trials ($\sum_t (\hat{\mu}_t - \mu_t)^2$), as a function of $\gamma$. Error bars show standard errors of the means over 500 trials. Mean square error for optimal $\gamma = 3$ is 473, compared to exact learning error 313 (lower line). Model parameters were same as in Figure 1.

more subtle changes in $\mu_t$ can miss detection, such as the third large shift in $\mu$. Figure 2(b) shows higher ACh ($V_t$) and NE ($\beta_t$) levels both correspond to fast learning, *ie* fast shifting of $\hat{\mu}_t$. However, whereas NE is a constant monitor of prediction errors and fluctuates accordingly with every data point, ACh falls smoothly and predictably, and only depends on the observations when global changes in the environment have been detected. Figure 2(b) shows ladle-shaped dependence of estimation error, $|\hat{\mu} - \mu|$, on the threshold value $\gamma$. For the particular setting of model parameters used here, learning is optimal for $\gamma$ around 3.

## 4 Differential Effects of Disrupting ACh and NE Signalling

The different roles of the NE ($\beta_t$) and ACh ($V_t$) can be teased apart by disrupting each and observing the subsequent effects on learning in our model. We will examine several different manipulations of $\beta_t$ and $V_t$ that disrupt normal learning, and relate the results to impairments observed in experimental manipulation of ACh or NE levels in animals. Of course, the complete experimental circumstances are far more complicated; we consider the general nature of the effects.

First, we simulate depletion of cortical NE by setting $\beta_t = 0$. An example is shown in Figure 3(a). By ruling out the possibility of $a_t = 1$, the system is unable to cope with abrupt, global changes in the world, *ie* when $\mu_t$ shifts. Mean error over 500 trials (same setting as in Figure 2(c)) without NE is 6027, more than an order of magnitude larger than full approximate learning (473) and exact learning (313). This is consistent with the large errors of similar magnitude in Figure 2(c) for very large $\gamma$, which effectively blocks the NE system from reporting global changes. However, as long as the underlying parameters remain the same, *ie* $\mu_t$ does not change greatly, the inference process functions normally, as we can see in the first 30 steps in Figure 3(a). These results are consistent with experimental observations: NE-lesioned animals are impaired in learning changes in reinforcement contingencies,[26,28] but have little difficulty doing previously learned discrimination tasks.[21]

We can also simulate depletion of cortical ACh by setting $V_t$ to a small constant value. Figure 3(b) shows severe damage is caused the learning algorithm, but the inference symptoms are distinct from NE depletion. Permanently small $V_t$ corresponds to over-confidence in estimates of $\hat{\mu}_t$, thus making adaptation of that estimate slow, similar to NE depletion. However, because the NE system is still intact, the system is able to detect when $x_t$ dramatically differs from the prediction (which is often, since $\hat{\mu}$ is slow to adapt and leaves little room for variance), and thus to base inference of $y_t$ directly on the bottom-up information $p[y_t|\mathbf{x}_t]$. Thus, inference is less impaired than learning, which has also been observed in

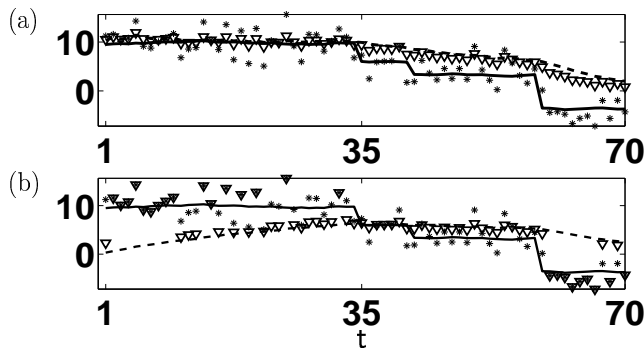

**Figure 3**: Disrupting NE and ACh signals. (a) NE signal set to 0. (b) ACh signal set to 0.16. $-$:$mu_t$, $--$:$\hat{\mu}$, $\nabla$:$\hat{y}_t$, $*$:projection of $\mathbf{x}_t$ into $y$ space. Learning of $\hat{m}u_t$ is poor in both manipulations, but inference in ACh-depletion is less impaired.

ACh-lesioned animals.[31] Moreover, the system exhibits a peculiar hesitancy in inference, *ie* constantly switching back and forth between relying on top-down estimate of $y_t$, based on $\mu_t$ and bottom-up estimate, based on $\boldsymbol{\xi}^T \boldsymbol{\Psi}^{-1} \mathbf{x}_t / \boldsymbol{\xi}^T \boldsymbol{\Psi}^{-1} \boldsymbol{\xi}$. This tendency is particularly severe when the new $\mu_t$ is similar to the previous one, which can be thought of as a form of interference. Interestingly, hippocampal cholinergic deafferentation in animals also bring about a stronger susceptibility to interference compared with controls.[10]

Saturation of ACh and NE are also easy to model, by setting $V_t$ and $\beta_t$ very high all the time. The effect of these two manipulations are similar, both cause the estimation of $\mu_t$ and inference of $y_t$ to base strongly on the observation $\mathbf{x}_t$ (data not shown). The performance decrements in the estimation of $\mu_t$ and inference about $y_t$ are functions of the output noise, $\Sigma_x, \sigma_y^2$ in our model, and do not worsen when there are global changes in contingencies. Unfortunately, directly relevant experimental data is scarce. Administration of cholinergic agonists in the cortex has failed to induce impairments in tasks with changing contingencies, consistent with our predictions. However, to our knowledge, cholinergic and noradrenergic agonists have not yet been administered in combination with systematic manipulation of variability in the predictive consequences of stimuli and so the validity of our predictions remains to be tested.

## 5   Discussion

We have suggested that ACh and NE report *expected* and *unexpected* uncertainty in representational learning and inference. As such, high levels of ACh and NE should both correspond to faster learning about the environment and enhancement of bottom-up processing in inference. However, whereas NE reports on dramatic changes, ACh has the subtler role of reporting on uncertainties in internal estimates.

We formalized these ideas in an adaptive factor analysis model. The model is adaptive in that the mean of the hidden variable is allowed to alter greatly from time to time, capturing the idea of a generally stable context which occasionally undergoes large changes, leading to substantial novelty in inputs. As exact learning is intractable, we proposed an approximate learning algorithm in which the roles for ACh and NE are clear, and demonstrated that it performs learning and inference competently. Moreover, by disrupting one or both of ACh and NE signalling systems, we showed that the two systems have interacting but distinct patterns of malfunctioning that qualitatively resemble experimental results in animal studies. There is no single collection of definitive experimental studies, and teasing apart the effects of NE and ACh is tricky, since they appear to share many properties. Our model helps understand why, and should also help with the design of experiments to clarify the relationship.

Of course, the adaptive factor analysis model is overly simple in many ways. In particular, it only considers one particular context; and so refers all the uncertainty to the parameters of that context. This is exactly the complement of our previous model,[6,35] which referred all the uncertainty to the choice of context rather than the parameters within each context. The main conceptual difference is that the idea that ACh reports on the latter form of contextual uncertainty sits ill with the data on how uncertainty boosts learning; this fits better within the present model. Given multiple contexts, which could formally be handled within the framework of a mixture model, the tricky issue is to decide whether the parameters of the current context have changed, or a new (or pre-existing) context has taken over. Exploring this is important work for the future. More generally, a thoroughly hierarchical and non-linear model is clearly required as at a minimum as a way of addressing some of the complexities of cortical inference.

**Acknowledgement**

We are very grateful to Zoubin Ghahramani and Maneesh Sahani for helpful discussions. Funding was from the Gatsby Charitable Foundation and the NSF.

# References

[1] Aston-Jones, G, Rajkowski, J, & Cohen, J (1999) *Biol Psychiatry* **46**:1309-1320.

[2] Carli, M, Robbins, TW, Evenden, JL, & Everitt, BJ (1983) *Behav Brain Res* **9**:361-80.

[3] Dalley, JW et al. (2001) *J Neurosci* **21**:4908-4914.

[4] Daw, ND, Kakade, S, & Dayan, P (2001) *Neural Networks* **15**:603-616.

[5] Dayan, P, Kakade, S, & Montague, PR (2000) In *NIPS 2000*:451-457.

[6] Dayan, P & Yu, A (2002) In *NIPS 2002*.

[7] Ghahramani, Z & Hinton, G (2000) *Neural Computation* **12**:831-64.

[8] Gil, Z, Conners, BW, & Amitai, Y (1997) *Neuron* **19**:679-86.

[9] Gu, Q (2002) *Neuroscience,* **111**:815-835.

[10] Hasselmo, ME (1995) *Behavioural Brain Research* **67**:1-27.

[11] Hasselmo, ME, Wyble, BP & Wallenstein, GV (1996) *Hippocampus* **6**:693-708.

[12] Hasselmo, ME & Cekic, M (1996) *Behavioural Brain Research* **79**: 153-161.

[13] Hasselmo, ME et al (1997) *J Neurophysiology* **78**:393-408.

[14] Holland, PC & Gallagher, M (1999) *Trends In Cognitive Sciences* **3**:65-73.

[15] Hsieh, CY, Cruikshank, SJ, & Metherate, R (2000) *Brain Research* **880**:51064.

[16] Kakade, S & Dayan, P (2002) *Psychological Review* **109**:533-544.

[17] Kimura, F, Fukuada, M, & Tsumoto, T (1999) *Eur. Jour. of Neurosci.* **11**:3597-3609.

[18] Kobayashi, M et al. (1999) *European Journal of Neuroscience* **12**:264-272.

[19] Mason, ST & Iversen, SD (1978) *Brain Res* **150**:135-48.

[20] McCormick, DA (1989) *Trends Neurosci* **12**:215-221.

[21] McGaughy, J, Sandstrom, M, *et al* (1997) *Behav Neurosci* **111**:646-52.

[22] McGaughy, J & Sarter, M (1998) *Behav Neurosci* **112**:1519-25.

[23] Minka, TP (2001) *A Family of Algorithms for Approximate Bayesian Inference.* PhD, MIT.

[24] Pearce, JM & Hall, G (1980) *Psychological Review* **87**:532-552.

[25] Rajkowski, J, Kubiak, P, & Aston-Jones, G (1994) *Brain Res Bull* **35**:607-16.

[26] Robbins, TW (1984) *Psychological Medicine* **14**:13-21.

[27] Robbins, TW, Everitt, BJ, & Cole, BJ (1985) *Physiological Psychology* **13**:127-150.

[28] Sara, SJ, Vankov, A, & Herve, A (1994) *Brain Res Bull* **35**:457-65.

[29] Sara, SJ (1998) *Comptes Rendus de l'Academie des Sciences Serie III* **321**:193-198.

[30] Sarter, M, Bruno, JP (1997) *Brain Research Reviews* **23**:28-46.

[31] Sarter, M, Holley, LA, & Matell, M (2000) In *SFN 2000* abstracts.

[32] Sullivan, RM (2001) *Ingegrative Physiological and Behavioral Science* **36**:293-307.

[33] Usher, M, et al. (1999) *Science* **5401**:549-554.

[34] Vankov, A, Herve-Minvielle, A, & Sara, SJ (1995) *Eur J Neurosci* **109**:903-911.

[35] Yu, A & Dayan, P (2002) *Neural Networks* **15**:719-730
